# PSVM: Parallelizing Support Vector Machines on Distributed Computers

**Edward Y. Chang,**[*] **Kaihua Zhu, Hao Wang, Hongjie Bai,**
Jian Li, Zhihuan Qiu, & Hang Cui
Google Research, Beijing, China

## Abstract

Support Vector Machines (SVMs) suffer from a widely recognized scalability problem in both memory use and computational time. To improve scalability, we have developed a parallel SVM algorithm (PSVM), which reduces memory use through performing a row-based, approximate matrix factorization, and which loads only essential data to each machine to perform parallel computation. Let $n$ denote the number of training instances, $p$ the reduced matrix dimension after factorization ($p$ is significantly smaller than $n$), and $m$ the number of machines. PSVM reduces the memory requirement from $\mathcal{O}(n^2)$ to $\mathcal{O}(np/m)$, and improves computation time to $\mathcal{O}(np^2/m)$. Empirical study shows PSVM to be effective. PSVM Open Source is available for download at http://code.google.com/p/psvm/.

## 1 Introduction

Let us examine the resource bottlenecks of SVMs in a binary classification setting to explain our proposed solution. Given a set of training data $\mathcal{X} = \{(\mathbf{x}_i, y_i) | \mathbf{x}_i \in \mathbf{R}^d\}_{i=1}^n$, where $\mathbf{x}_i$ is an observation vector, $y_i \in \{-1, 1\}$ is the class label of $\mathbf{x}_i$, and $n$ is the size of $\mathcal{X}$, we apply SVMs on $\mathcal{X}$ to train a binary classifier. SVMs aim to search a hyperplane in the *Reproducing Kernel Hilbert Space* (RKHS) that maximizes the margin between the two classes of data in $\mathcal{X}$ with the smallest training error (Vapnik, 1995). This problem can be formulated as the following quadratic optimization problem:

$$\min \quad \mathcal{P}(\mathbf{w}, b, \boldsymbol{\xi}) = \frac{1}{2}\|\mathbf{w}\|_2^2 + C\sum_{i=1}^n \xi_i \tag{1}$$

$$s.t. \quad 1 - y_i(\mathbf{w}^T \boldsymbol{\phi}(\mathbf{x}_i) + b) \leq \xi_i, \quad \xi_i > 0,$$

where $\mathbf{w}$ is a weighting vector, $b$ is a threshold, $C$ a regularization hyperparameter, and $\boldsymbol{\phi}(\cdot)$ a basis function which maps $\mathbf{x}_i$ to an RKHS space. The decision function of SVMs is $f(\mathbf{x}) = \mathbf{w}^T \boldsymbol{\phi}(\mathbf{x}) + b$, where the $\mathbf{w}$ and $b$ are attained by solving $\mathcal{P}$ in (1). The optimization problem in (1) is the primal formulation of SVMs. It is hard to solve $\mathcal{P}$ directly, partly because the explicit mapping via $\boldsymbol{\phi}(\cdot)$ can make the problem intractable and partly because the mapping function $\boldsymbol{\phi}(\cdot)$ is often unknown. The method of *Lagrangian multipliers* is thus introduced to transform the primal formulation into the dual one

$$\min \quad \mathcal{D}(\boldsymbol{\alpha}) = \frac{1}{2}\boldsymbol{\alpha}^T \mathbf{Q} \boldsymbol{\alpha} - \boldsymbol{\alpha}^T \mathbf{1} \tag{2}$$

$$s.t. \quad \mathbf{0} \leq \boldsymbol{\alpha} \leq \mathbf{C}, \ \mathbf{y}^T \boldsymbol{\alpha} = 0,$$

where $[\mathbf{Q}]_{ij} = y_i y_j \boldsymbol{\phi}^T(\mathbf{x}_i) \boldsymbol{\phi}(\mathbf{x}_j)$, and $\boldsymbol{\alpha} \in \mathbf{R}^n$ is the Lagrangian multiplier variable (or dual variable). The weighting vector $\mathbf{w}$ is related with $\boldsymbol{\alpha}$ in $\mathbf{w} = \sum_{i=1}^n \alpha_i \boldsymbol{\phi}(\mathbf{x}_i)$.

---

[*]This work was initiated in 2005 when the author was a professor at UCSB.

The dual formulation $\mathcal{D}(\boldsymbol{\alpha})$ requires an inner product of $\phi(\mathbf{x}_i)$ and $\phi(\mathbf{x}_j)$. SVMs utilize the *kernel trick* by specifying a kernel function to define the inner-product $K(\mathbf{x}_i, \mathbf{x}_j) = \phi^T(\mathbf{x}_i)\phi(\mathbf{x}_j)$. We thus can rewrite $[\mathbf{Q}]_{ij}$ as $y_i y_j K(\mathbf{x}_i, \mathbf{x}_j)$. When the given kernel function $K$ is psd (positive semi-definite), the dual problem $\mathcal{D}(\boldsymbol{\alpha})$ is a convex Quadratic Programming (QP) problem with linear constraints, which can be solved via the *Interior-Point method* (IPM) (Mehrotra, 1992). Both the computational and memory bottlenecks of the SVM training are the IPM solver to the dual formulation of SVMs in (2).

Currently, the most effective IPM algorithm is the primal-dual IPM (Mehrotra, 1992). The principal idea of the primal-dual IPM is to remove inequality constraints using a barrier function and then resort to the iterative Newton's method to solve the KKT linear system related to the Hessian matrix $\mathbf{Q}$ in $\mathcal{D}(\boldsymbol{\alpha})$. The computational cost is $O(n^3)$ and the memory usage $O(n^2)$.

In this work, we propose a parallel SVM algorithm (PSVM) to reduce memory use and to parallelize both data loading and computation. Given $n$ training instances each with $d$ dimensions, PSVM first loads the training data in a round-robin fashion onto $m$ machines. The memory requirement per machine is $\mathcal{O}(nd/m)$. Next, PSVM performs a parallel row-based Incomplete Cholesky Factorization (ICF) on the loaded data. At the end of parallel ICF, each machine stores only a fraction of the factorized matrix, which takes up space of $\mathcal{O}(np/m)$, where $p$ is the column dimension of the factorized matrix. (Typically, $p$ can be set to be about $\sqrt{n}$ without noticeably degrading training accuracy.) PSVM reduces memory use of IPM from $\mathrm{O}(n^2)$ to $O(np/m)$, where $p/m$ is much smaller than $n$. PSVM then performs parallel IPM to solve the quadratic optimization problem in (2). The computation time is improved from about $\mathcal{O}(n^2)$ of a decomposition-based algorithm (e.g., SVMLight (Joachims, 1998), LIBSVM (Chang & Lin, 2001), SMO (Platt, 1998), and SimpleSVM (Vishwanathan et al., 2003)) to $\mathcal{O}(np^2/m)$. This work's main contributions are: (1) PSVM achieves memory reduction and computation speedup via a parallel ICF algorithm and parallel IPM. (2) PSVM handles kernels (in contrast to other algorithmic approaches (Joachims, 2006; Chu et al., 2006)). (3) We have implemented PSVM on our parallel computing infrastructures. PSVM effectively speeds up training time for large-scale tasks while maintaining high training accuracy.

PSVM is a practical, parallel approximate implementation to speed up SVM training on today's distributed computing infrastructures for dealing with Web-scale problems. What we do **not** claim are as follows: (1) We make no claim that PSVM is the sole solution to speed up SVMs. Algorithmic approaches such as (Lee & Mangasarian, 2001; Tsang et al., 2005; Joachims, 2006; Chu et al., 2006) can be more effective when memory is not a constraint or kernels are not used. (2) We do not claim that the algorithmic approach is the only avenue to speed up SVM training. Data-processing approaches such as (Graf et al., 2005) can divide a serial algorithm (e.g., LIBSVM) into subtasks on subsets of training data to achieve good speedup. (Data-processing and algorithmic approaches complement each other, and can be used together to handle large-scale training.)

## 2 PSVM Algorithm

The key step of PSVM is parallel ICF (PICF). Traditional column-based ICF (Fine & Scheinberg, 2001; Bach & Jordan, 2005) can reduce computational cost, but the initial memory requirement is $O(np)$, and hence not practical for very large data set. PSVM devises parallel row-based ICF (PICF) as its initial step, which loads training instances onto parallel machines and performs factorization simultaneously on these machines. Once PICF has loaded $n$ training data distributedly on $m$ machines, and reduced the size of the kernel matrix through factorization, IPM can be solved on parallel machines simultaneously. We present PICF first, and then describe how IPM takes advantage of PICF.

### 2.1 Parallel ICF

ICF can approximate $Q$ ($Q \in R^{n \times n}$) by a smaller matrix $H$ ($H \in R^{n \times p}, p \ll n$), i.e., $Q \approx HH^T$. ICF, together with SMW (the *Sherman-Morrison-Woodbury formula*), can greatly reduce the computational complexity in solving an $n \times n$ linear system. The work of (Fine & Scheinberg, 2001) provides a theoretical analysis of how ICF influences the optimization problem in Eq.(2). The authors proved that the error of the optimal objective value introduced by ICF is bounded by $C^2 l \epsilon / 2$, where $C$ is the hyperparameter of SVM, $l$ is the number of support vectors, and $\epsilon$ is the bound of

---

**Algorithm 1** Row-based PICF

---

    **Input**: $n$ training instances; $p$: rank of ICF matrix $H$; $m$: number of machines
    **Output**: $H$ distributed on $m$ machines
    **Variables**:
    $\mathbf{v}$: fraction of the diagonal vector of $Q$ that resides in local machine
    $k$: iteration number;
    $\mathbf{x}_i$: the $i^{th}$ training instance
    $M$: machine index set, $M = \{0, 1, \dots, m-1\}$
    $I_c$: row-index set on machine $c$ $(c \in M)$, $I_c = \{c, c+m, c+2m, \dots\}$
1: **for** $i = 0$ to $n-1$ **do**
2:      Load $\mathbf{x}_i$ into machine $i\,modulo\,m$.
3: **end for**
4: $k \leftarrow 0$; $H \leftarrow 0$; $\mathbf{v} \leftarrow$ the fraction of the diagonal vector of $Q$ that resides in local machine. $(\mathbf{v}(i)(i \in I_m)$ can be obtained from $\mathbf{x}_i$)
5: Initialize *master* to be machine 0.
6: **while** $k < p$ **do**
7:      Each machine $c \in M$ selects its local pivot value, which is the largest element in $\mathbf{v}$:

$$\mathbf{lpv}_{k,c} = \max_{i \in I_c} \mathbf{v}(i).$$

       and records the local pivot index, the row index corresponds to $\mathbf{lpv}_{k,c}$:

$$\mathbf{lpi}_{k,c} = arg \max_{i \in I_c} \mathbf{v}(i).$$

8:      Gather $\mathbf{lpv}_{k,c}$'s and $\mathbf{lpi}_{k,c}$'s $(c \in M)$ to *master*.
9:      The *master* selects the largest local pivot value as global pivot value $\mathbf{gpv}_k$ and records in $i_k$, row index corresponding to the global pivot value.

$$\mathbf{gpv}_k = \max_{c \in M} \mathbf{lpv}_{k,c}.$$

10:     The *master* broadcasts $\mathbf{gpv}_k$ and $i_k$.
11:     Change *master* to machine $i_k \% m$.
12:     Calculate $H(i_k, k)$ according to (3) on *master*.
13:     The *master* broadcasts the pivot instance $\mathbf{x}_{i_k}$ and the pivot row $H(i_k, :)$. (Only the first $k+1$ values of the pivot row need to be broadcast, since the remainder are zeros.)
14:     Each machine $c \in M$ calculates its part of the $k^{th}$ column of $H$ according to (4).
15:     Each machine $c \in M$ updates $\mathbf{v}$ according to (5).
16:     $k \leftarrow k+1$
17: **end while**

---

ICF approximation (i.e. $tr(Q - HH^T) < \epsilon$). Experimental results in Section 3 show that when $p$ is set to $\sqrt{n}$, the error can be negligible.

Our row-based parallel ICF (PICF) works as follows: Let vector $\mathbf{v}$ be the diagonal of $Q$ and suppose the pivots (the largest diagonal values) are $\{i_1, i_2, \dots, i_k\}$, the $k^{th}$ iteration of ICF computes three equations:

$$H(i_k, k) = \sqrt{\mathbf{v}(i_k)} \tag{3}$$

$$H(J_k, k) = (Q(J_k, k) - \sum_{j=1}^{k-1} H(J_k, j)H(i_k, j))/H(i_k, k) \tag{4}$$

$$\mathbf{v}(J_k) = \mathbf{v}(J_k) - H(J_k, k)^2, \tag{5}$$

where $J_k$ denotes the complement of $\{i_1, i_2, \dots, i_k\}$. The algorithm iterates until the approximation of $Q$ by $H_k H_k^T$ (measured by $trace(Q - H_k H_k^T)$) is satisfactory, or the predefined maximum iterations (or say, the desired rank of the ICF matrix) $p$ is reached.

As suggested by G. Golub, a parallelized ICF algorithm can be obtained by constraining the parallelized Cholesky Factorization algorithm, iterating at most $p$ times. However, in the proposed algorithm (Golub & Loan, 1996), matrix $H$ is distributed by columns in a round-robin way on $m$ machines (hence we call it column-based parallelized ICF). Such column-based approach is optimal for the single-machine setting, but cannot gain full benefit from parallelization for two major reasons:

**1**. Large memory requirement. All training data are needed for each machine to calculate $Q(J_k, k)$. Therefore, each machine must be able to store a local copy of the training data.

**2**. Limited parallelizable computation. Only the inner product calculation $(\sum_{j=1}^{k-1} H(J_k, j) H(i_k, j))$ in (4) can be parallelized. The calculation of pivot selection, the summation of local inner product result, column calculation in (4), and the vector update in (5) must be performed on one single machine.

To remedy these shortcomings of the column-based approach, we propose a row-based approach to parallelize ICF, which we summarize in Algorithm 1. Our row-based approach starts by initializing variables and loading training data onto $m$ machines in a round-robin fashion (Steps 1 to 5). The algorithm then performs the ICF main loop until the termination criteria are satisfied (e.g., the rank of matrix $H$ reaches $p$). In the main loop, PICF performs five tasks in each iteration $k$:

- Distributedly find a pivot, which is the largest value in the diagonal $\mathbf{v}$ of matrix $Q$ (steps 7 to 10). Notice that PICF computes only needed elements in $Q$ from training data, and it does not store $Q$.
- Set the machine where the pivot resides as the *master* (step 11).
- On the *master*, PICF calculates $H(i_k, k)$ according to (3) (step 12).
- The *master* then broadcasts the pivot instance $\mathbf{x}_{i_k}$ and the pivot row $H(i_k, :)$ (step 13).
- Distributedly compute (4) and (5) (steps 14 and 15).

At the end of the algorithm, $H$ is stored distributedly on $m$ machines, ready for parallel IPM (presented in the next section). PICF enjoys three advantages: parallel memory use ($\mathcal{O}(np/m)$), parallel computation ($\mathcal{O}(p^2 n/m)$), and low communication overhead ($\mathcal{O}(p^2 \log(m))$). Particularly on the communication overhead, its fraction of the entire computation time shrinks as the problem size grows. We will verify this in the experimental section. This pattern permits a larger problem to be solved on more machines to take advantage of parallel memory use and computation.

## 2.2 Parallel IPM

As mentioned in Section 1, the most effective algorithm to solve a constrained QP problem is the primal-dual IPM. For detailed description and notations of IPM, please consult (Boyd, 2004; Mehrotra, 1992). For the purpose of SVM training, IPM boils down to solving the following equations in the Newton step iteratively.

$$\triangle \boldsymbol{\lambda} = -\boldsymbol{\lambda} + \text{vec}\left(\frac{1}{t(C - \alpha_i)}\right) + \text{diag}(\frac{\lambda_i}{C - \alpha_i})\triangle \mathbf{x} \tag{6}$$

$$\triangle \boldsymbol{\xi} = -\boldsymbol{\xi} + \text{vec}\left(\frac{1}{t\alpha_i}\right) - \text{diag}(\frac{\xi_i}{\alpha_i})\triangle \mathbf{x} \tag{7}$$

$$\triangle \nu = \frac{\mathbf{y}^T \boldsymbol{\Sigma}^{-1} \mathbf{z} + \mathbf{y}^T \boldsymbol{\alpha}}{\mathbf{y}^T \boldsymbol{\Sigma}^{-1} \mathbf{y}} \tag{8}$$

$$D = \mathbf{diag}(\frac{\xi_i}{\alpha_i} + \frac{\lambda_i}{C - \alpha_i}) \tag{9}$$

$$\triangle \mathbf{x} = \boldsymbol{\Sigma}^{-1}(\mathbf{z} - \mathbf{y}\triangle \nu), \tag{10}$$

where $\boldsymbol{\Sigma}$ and $\mathbf{z}$ depend only on $[\boldsymbol{\alpha}, \boldsymbol{\lambda}, \boldsymbol{\xi}, \boldsymbol{\nu}]$ from the last iteration as follows:

$$\boldsymbol{\Sigma} = \mathbf{Q} + \mathbf{diag}(\frac{\xi_i}{\alpha_i} + \frac{\lambda_i}{C - \alpha_i}) \tag{11}$$

$$\mathbf{z} = -\mathbf{Q}\boldsymbol{\alpha} + \mathbf{1}_n - \nu\mathbf{y} + \frac{1}{t}\text{vec}(\frac{1}{\alpha_i} - \frac{1}{C - \alpha_i}). \tag{12}$$

The computation bottleneck is on matrix inverse, which takes place on $\boldsymbol{\Sigma}$ for solving $\triangle \nu$ in (8) and $\triangle \mathbf{x}$ in (10). Equation (11) shows that $\boldsymbol{\Sigma}$ depends on $Q$, and we have shown that $Q$ can be approximated through PICF by $HH^T$. Therefore, the bottleneck of the Newton step can be sped up from $\mathcal{O}(n^3)$ to $\mathcal{O}(p^2 n)$, and be parallelized to $\mathcal{O}(p^2 n/m)$.

**Distributed Data Loading**

To minimize both storage and communication cost, PIPM stores data distributedly as follows:

- *Distribute matrix data.* $H$ is distributedly stored at the end of PICF.

- *Distribute $n \times 1$ vector data.* All $n \times 1$ vectors are distributed in a round-robin fashion on $m$ machines. These vectors are $\mathbf{z}, \boldsymbol{\alpha}, \boldsymbol{\xi}, \boldsymbol{\lambda}, \Delta\mathbf{z}, \Delta\boldsymbol{\alpha}, \Delta\boldsymbol{\xi}$, and $\Delta\boldsymbol{\lambda}$.

- *Replicate global scalar data.* Every machine caches a copy of global data including $\nu$, $t$, $n$, and $\Delta\nu$. Whenever a scalar is changed, a broadcast is required to maintain global consistency.

**Parallel Computation of $\triangle\nu$**

Rather than walking through all equations, we describe how PIPM solves (8), where $\Sigma^{-1}$ appears twice. An interesting observation is that parallelizing $\Sigma^{-1}z$ (or $\Sigma^{-1}y$) is simpler than parallelizing $\Sigma^{-1}$. Let us explain how parallelizing $\Sigma^{-1}z$ works, and parallelizing $\Sigma^{-1}y$ can follow suit.

According to SMW (the *Sherman-Morrison-Woodbury formula*), we can write $\Sigma^{-1}z$ as

$$
\begin{aligned}
\Sigma^{-1}z &= (D+Q)^{-1}z \approx (D+HH^T)^{-1}z \\
&= D^{-1}z - D^{-1}H(I + H^T D^{-1} H)^{-1} H^T D^{-1} z \\
&= D^{-1}z - D^{-1}H(GG^T)^{-1} H^T D^{-1} z.
\end{aligned}
$$

$\Sigma^{-1}z$ can be computed in four steps:

**1**. Compute $D^{-1}z$. $D$ can be derived from locally stored vectors, following (9). $D^{-1}z$ is a $n \times 1$ vector, and can be computed locally on each of the $m$ machines.

**2**. Compute $t_1 = H^T D^{-1} z$. Every machine stores some rows of H and their corresponding part of $D^{-1}z$. This step can be computed locally on each machine. The results are sent to the *master* (which can be a randomly picked machine for all PIPM iterations) to aggregate into $t_1$ for the next step.

**3**. Compute $(GG^T)^{-1}t_1$. This step is completed on the *master*, since it has all the required data. $G$ can be obtained from $H$ in a straightforward manner as shown in SMW. Computing $t_2 = (GG^T)^{-1}t_1$ is equivalent to solving the linear equation system $t_1 = (GG^T)t_2$. PIPM first solves $t_1 = Gy_0$, then $y_0 = G^T t2$. Once it has obtained $y_0$, PIPM can solve $G^T t_2 = y_0$ to obtain $t_2$. The *master* then broadcasts $t_2$ to all machines.

**4**. Compute $D^{-1}Ht_2$ All machines have a copy of $t_2$, and can compute $D^{-1}Ht_2$ locally to solve for $\Sigma^{-1}z$.

Similarly, $\Sigma^{-1}y$ can be computed at the same time. Once we have obtained both, we can solve $\Delta\nu$ according to (8).

## 2.3 Computing $b$ and Writing Back

When the IPM iteration stops, we have the value of $\alpha$ and hence the classification function

$$
f(x) = \sum_{i=1}^{N_s} \alpha_i y_i \mathbf{k}(s_i, x) + b
$$

Here $N_s$ is the number of support vectors and $s_i$ are support vectors. In order to complete this classification function, $b$ must be computed. According to the SVM model, given a support vector $s$, we obtain one of the two results for $f(s)$: $f(s) = +1$, if $y_s = +1$, or $f(s) = -1$, if $y_s = -1$.

In practice, we can select $M$, say $1,000$, support vectors and compute the average of the $b_s$ in parallel using MapReduce (Dean & Ghemawat, 2004).

## 3 Experiments

We conducted experiments on PSVM to evaluate its 1) class-prediction accuracy, 2) scalability on large datasets, and 3) overheads. The experiments were conducted on up to 500 machines in our data center. Not all machines are identically configured; however, each machine is configured with a CPU faster than 2GHz and memory larger than 4GBytes.

Table 1: Class-prediction Accuracy with Different $p$ Settings.

| dataset | samples (train/test) | LIBSVM | $p = n^{0.1}$ | $p = n^{0.2}$ | $p = n^{0.3}$ | $p = n^{0.4}$ | $p = n^{0.5}$ |
|---|---|---|---|---|---|---|---|
| *svmguide1* | $3,089/4,000$ | 0.9608 | 0.6563 | 0.9 | 0.917 | 0.9495 | 0.9593 |
| *mushrooms* | $7,500/624$ | 1 | 0.9904 | 0.9920 | 1 | 1 | 1 |
| *news20* | $18,000/1,996$ | 0.7835 | 0.6949 | 0.6949 | 0.6969 | 0.7806 | 0.7811 |
| *Image* | $199,957/84,507$ | 0.849 | 0.7293 | 0.7210 | 0.8041 | 0.8121 | 0.8258 |
| *CoverType* | $522,910/58,102$ | 0.9769 | 0.9764 | 0.9762 | 0.9766 | 0.9761 | 0.9766 |
| *RCV* | $781,265/23,149$ | 0.9575 | 0.8527 | 0.8586 | 0.8616 | 0.9065 | 0.9264 |

## 3.1 Class-prediction Accuracy

PSVM employs PICF to approximate an $n \times n$ kernel matrix $Q$ with an $n \times p$ matrix $H$. This experiment evaluated how the choice of $p$ affects class-prediction accuracy. We set $p$ of PSVM to $n^t$, where $t$ ranges from 0.1 to 0.5 incremented by 0.1, and compared its class-prediction accuracy with that achieved by LIBSVM. The first two columns of Table 1 enumerate the datasets and their sizes with which we experimented. We use Gaussian kernel, and select the best $C$ and $\sigma$ for LIBSVM and PSVM, respectively. For *CoverType* and *RCV*, we loosed the terminate condition (set -e 1, default 0.001) and used shrink heuristics (set -h 0) to make LIBSVM terminate within several days. The table shows that when $t$ is set to 0.5 (or $p = \sqrt{n}$), the class-prediction accuracy of PSVM approaches that of LIBSVM.

We compared only with LIBSVM because it is arguably the best open-source SVM implementation in both accuracy and speed. Another possible candidate is CVM (Tsang et al., 2005). Our experimental result on the *CoverType* dataset outperforms the result reported by CVM on the same dataset in both accuracy and speed. Moreover, CVM's training time has been shown unpredictable by (Loosli & Canu, 2006), since the training time is sensitive to the selection of stop criteria and hyper-parameters. For how we position PSVM with respect to other related work, please refer to our disclaimer in the end of Section 1.

## 3.2 Scalability

For scalability experiments, we used three large datasets. Table 2 reports the speedup of PSVM on up to $m = 500$ machines. Since when a dataset size is large, a single machine cannot store the factorized matrix $H$ in its local memory, we cannot obtain the running time of PSVM on one machine. We thus used 10 machines as the baseline to measure the speedup of using more than 10 machines. To quantify speedup, we made an assumption that the speedup of using 10 machines is 10, compared to using one machine. This assumption is reasonable for our experiments, since PSVM does enjoy linear speedup when the number of machines is up to 30.

Table 2: Speedup ($p$ is set to $\sqrt{n}$); LIBSVM training time is reported on the last row for reference.

| Machines | Image (200k) | | | CoverType (500k) | | | RCV (800k) | | |
|---|---|---|---|---|---|---|---|---|---|
| | Time (s) | | Speedup | Time (s) | | Speedup | Time (s) | | Speedup |
| 10 | $1,958$ | (9) | $10^*$ | $16,818$ | (442) | $10^*$ | $45,135$ | (1373) | $10^*$ |
| 30 | 572 | (8) | 34.2 | $5,591$ | (10) | 30.1 | $12,289$ | (98) | 36.7 |
| 50 | 473 | (14) | 41.4 | $3,598$ | (60) | 46.8 | $7,695$ | (92) | 58.7 |
| 100 | 330 | (47) | 59.4 | $2,082$ | (29) | 80.8 | $4,992$ | (34) | 90.4 |
| 150 | 274 | (40) | 71.4 | $1,865$ | (93) | 90.2 | $3,313$ | (59) | 136.3 |
| 200 | 294 | (41) | 66.7 | $1,416$ | (24) | 118.7 | $3,163$ | (69) | 142.7 |
| 250 | 397 | (78) | 49.4 | $1,405$ | (115) | 119.7 | $2,719$ | (203) | 166.0 |
| 500 | 814 | (123) | 24.1 | $1,655$ | (34) | 101.6 | $2,671$ | (193) | 169.0 |
| LIBSVM | $4,334$ | NA | NA | $28,149$ | NA | NA | $184,199$ | NA | NA |

We trained PSVM three times for each dataset-$m$ combination. The speedup reported in the table is the average of three runs with standard deviation provided in brackets. The observed variance in speedup was caused by the variance of machine loads, as all machines were shared with other tasks

running on our data centers. We can observe in Table 2 that the larger is the dataset, the better is the speedup. Figures 1(a), (b) and (c) plot the speedup of *Image*, *CoverType*, and *RCV*, respectively. All datasets enjoy a linear speedup when the number of machines is moderate. For instance, PSVM achieves linear speedup on *RCV* when running on up to around 100 machines. PSVM scales well till around 250 machines. After that, adding more machines receives diminishing returns. This result led to our examination on the overheads of PSVM, presented next.

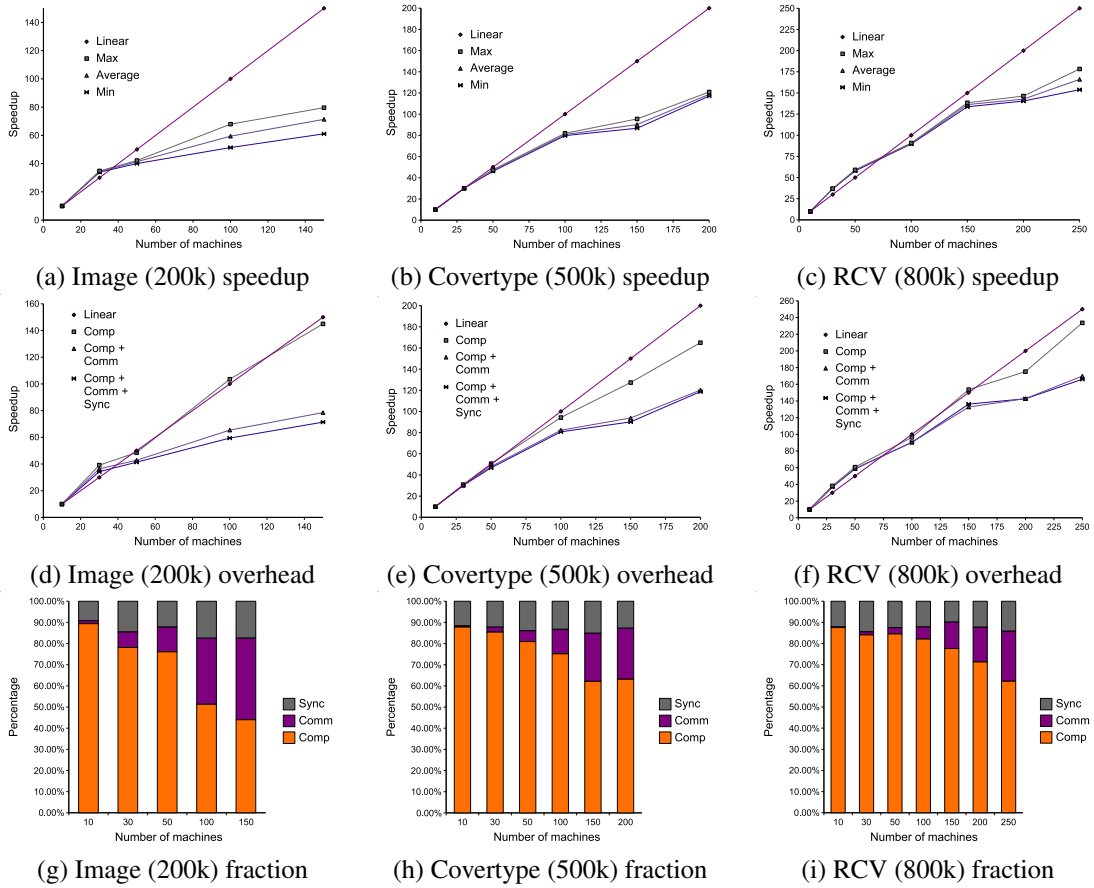

Figure 1: Speedup and Overheads of Three Datasets.

## 3.3 Overheads

PSVM cannot achieve linear speedup when the number of machines continues to increase beyond a data-size-dependent threshold. This is expected due to communication and synchronization overheads. Communication time is incurred when message passing takes place between machines. Synchronization overhead is incurred when the *master* machine waits for task completion on the slowest machine. (The *master* could wait forever if a child machine fails. We have implemented a checkpoint scheme to deal with this issue.)

The running time consists of three parts: computation (Comp), communication (Comm), and synchronization (Sync). Figures 1(d), (e) and (f) show how Comm and Sync overheads influence the speedup curves. In the figures, we draw on the top the computation only line (Comp), which approaches the linear speedup line. Computation speedup can become sublinear when adding machines beyond a threshold. This is because the computation bottleneck of the unparallelizable step 12 in Algorithm 1 (which computation time is $\mathcal{O}(p^2)$). When $m$ is small, this bottleneck is insignificant in the total computation time. According to the Amdahl's law; however, even a small fraction of unparallelizable computation can cap speedup. Fortunately, the larger the dataset is, the smaller is this unparallelizable fraction, which is $\mathcal{O}(m/n)$. Therefore, more machines (larger $m$) can be employed for larger datasets (larger $n$) to gain speedup.

When communication overhead or synchronization overhead is accounted for (the Comp + Comm line and the Comp + Comm + Sync line), the speedup deteriorates. Between the two overheads, the synchronization overhead does not impact speedup as much as the communication overhead does. Figures 1(g), (h), and (i) present the percentage of Comp, Comm, and Sync in total running time. The synchronization overhead maintains about the same percentage when $m$ increases, whereas the percentage of communication overhead grows with $m$. As mentioned in Section 2.1, the communication overhead is $\mathcal{O}(p^2 \log(m))$, growing sub-linearly with $m$. But since the computation time per node decreases as $m$ increases, the fraction of the communication overhead grows with $m$. Therefore, PSVM must select a proper $m$ for a training task to maximize the benefit of parallelization.

## 4   Conclusion

In this paper, we have shown how SVMs can be parallelized to achieve scalable performance. PSVM distributedly loads training data on parallel machines, reducing memory requirement through approximate factorization on the kernel matrix. PSVM solves IPM in parallel by cleverly arranging computation order. We have made PSVM open source at http://code.google.com/p/psvm/.

## Acknowledgement

The first author is partially supported by NSF under Grant Number IIS-0535085.

## References

Bach, F. R., & Jordan, M. I. (2005). Predictive low-rank decomposition for kernel methods. *Proceedings of the 22nd International Conference on Machine Learning*.

Boyd, S. (2004). *Convex optimization*. Cambridge University Press.

Chang, C.-C., & Lin, C.-J. (2001). *LIBSVM: a library for support vector machines*. Software available at http://www.csie.ntu.edu.tw/ cjlin/libsvm.

Chu, C.-T., Kim, S. K., Lin, Y.-A., Yu, Y., Bradski, G., Ng, A. Y., & Olukotun, K. (2006). Map reduce for machine learning on multicore. *NIPS*.

Dean, J., & Ghemawat, S. (2004). Mapreduce: Simplified data processing on large clusters. *OSDI'04: Symposium on Operating System Design and Implementation*.

Fine, S., & Scheinberg, K. (2001). Efficient svm training using low-rank kernel representations. *Journal of Machine Learning Research*, 2, 243–264.

Ghemawat, S., Gobioff, H., & Leung, S.-T. (2003). The google file system. *19th ACM Symposium on Operating Systems Principles*.

Golub, G. H., & Loan, C. F. V. (1996). *Matrix computations*. Johns Hopkins University Press.

Graf, H. P., Cosatto, E., Bottou, L., Dourdanovic, I., & Vapnik, V. (2005). Parallel support vector machines: The cascade svm. In *Advances in neural information processing systems 17*, 521–528.

Joachims, T. (1998). Making large-scale svm learning practical. *Advances in Kernel Methods - Support Vector Learning*.

Joachims, T. (2006). Training linear svms in linear time. *ACM KDD*, 217–226.

Lee, Y.-J., & Mangasarian, O. L. (2001). Rsvm: Reduced support vector machines. *First SIAM International Conference on Data Mining*. Chicago.

Loosli, G., & Canu, S. (2006). *Comments on the core vector machines: Fast svm training on very large data sets* (Technical Report).

Mehrotra, S. (1992). On the implementation of a primal-dual interior point method. *SIAM J. Optimization*, 2.

Platt, J. (1998). *Sequential minimal optimization: A fast algorithm for training support vector machines* (Technical Report MSR-TR-98-14). Microsoft Research.

Tsang, I. W., Kwok, J. T., & Cheung, P.-M. (2005). Core vector machines: Fast svm training on very large data sets. *Journal of Machine Learning Research*, 6, 363–392.

Vapnik, V. (1995). *The nature of statistical learning theory*. New York: Springer.

Vishwanathan, S., Smola, A. J., & Murty, M. N. (2003). Simplesvm. *ICML*.

